# The Fixed Points of Off-Policy TD

**J. Zico Kolter**
Computer Science and Artificial Intelligence Laboratory
Massachusetts Institute of Technology
Cambridge, MA 02139
kolter@csail.mit.edu

## Abstract

Off-policy learning, the ability for an agent to learn about a policy other than the one it is following, is a key element of Reinforcement Learning, and in recent years there has been much work on developing Temporal Different (TD) algorithms that are guaranteed to converge under off-policy sampling. It has remained an open question, however, whether anything can be said a priori about the *quality* of the TD solution when off-policy sampling is employed with function approximation. In general the answer is no: for arbitrary off-policy sampling the error of the TD solution can be unboundedly large, even when the approximator can represent the true value function well. In this paper we propose a novel approach to address this problem: we show that by considering a certain *convex subset* of off-policy distributions we can indeed provide guarantees as to the solution quality similar to the on-policy case. Furthermore, we show that we can efficiently project on to this convex set using only samples generated from the system. The end result is a novel TD algorithm that has approximation guarantees even in the case of off-policy sampling and which empirically outperforms existing TD methods.

## 1 Introduction

In temporal prediction tasks, Temporal Difference (TD) learning provides a method for learning long-term expected rewards (the "value function") using only trajectories from the system. The algorithm is ubiquitous in Reinforcement Learning, and there has been a great deal of work studying the convergence properties of the algorithm. It is well known that for a tabular value function representation, TD converges to the true value function [3, 4]. For linear function approximation with on-policy sampling (i.e., when the states are drawn from the stationary distribution of the policy we are trying to evaluate), the algorithm converges to a well-known fixed point that is guaranteed to be close to the optimal projection of the true value function [17]. When states are sampled off-policy, standard TD may diverge when using linear function approximation [1], and this has led in recent years to a number of modified TD algorithms that are guaranteed to convergence even in the presence of off-policy sampling [16, 15, 9, 10].

Of equal importance, however, is the actual *quality* of the TD solution under off-policy sampling. Previous work, as well as an example we present in this paper, show that in general little can be said about this question: the solution found by TD can be arbitrarily poor in the case of off-policy sampling, even when the true value function is well-approximated by a linear basis. Pursing a slightly different approach, other recent work has looked at providing problem dependent bounds, which use problem-specific matrices to obtain tighter bounds than previous approaches [19]; these bounds can apply to the off-policy setting, but depend on problem data, and will still fail to provide a reasonable bound in the cases mentioned above where the off-policy approximation is arbitrarily poor. Indeed, a long-standing open question in Reinforcement Learning is whether any a priori guarantees can be made about the solution quality for off-policy methods using function approximation.

In this paper we propose a novel approach that addresses this question: we present an algorithm that looks for a *subset* of off-policy sampling distributions where a certain relaxed contraction property

holds; for distributions in this set, we show that it is indeed possible to obtain error bounds on the solution quality similar to those for the on-policy case. Furthermore, we show that this set of feasible off-policy sampling distributions is convex, representable via a linear matrix inequality (LMI), and we demonstrate how the set can be approximated and projected onto efficiently in the finite sample setting. The resulting method, which we refer to as TD with distribution optimization (TD-DO), is thus able to guarantee a good approximation to the best possible projected value function, even for off-policy sampling. In simulations we show that the algorithm can improve significantly over standard off-policy TD.

## 2    Preliminaries and Background

A Markov chain is a tuple, $(S, P, R, \gamma)$, where $S$ is a set of states, $P : S \times S \to \mathbb{R}_+$ is a transition probability function, $R : S \to \mathbb{R}$ is a reward function, and $\gamma \in [0, 1)$ is a discount factor. For simplicity of presentation we will assume the state space is countable, and so can be indexed by the set $S = \{1, \ldots, n\}$, which allows us to use matrix rather than operator notation. The *value function* for a Markov chain, $V : S \to \mathbb{R}$ maps states to their long term discounted sum of rewards, and is defined as $V(s) = \mathbf{E}\left[\sum_{t=0}^{\infty} \gamma^t R(s_t)|s_0 = s\right]$. The value function may also be expressed via Bellman's equation (in vector form)

$$V = R + \gamma P V \tag{1}$$

where $R, V \in \mathbb{R}^n$ represent vectors of all rewards and values respectively, and $P \in \mathbb{R}^{n \times n}$ is a matrix of probability transitions $P_{ij} = P(s' = j|s = i)$.

In linear function approximation, the value function is approximated as a linear combination of some features describing the state: $V(s) \approx w^T \phi(s)$, where $w \in \mathbb{R}^k$ is a vector of parameters, and $\phi : S \to \mathbb{R}^k$ is a function mapping states to $k$-dimensional feature vectors; or, again using vector notation, $V \approx \Phi w$, where $\Phi \in \mathbb{R}^{n \times k}$ is a matrix of all feature vectors. The TD solution is a fixed point of the Bellman operator followed by a projection, i.e.,

$$\Phi w_D^\star = \Pi_D(R + \gamma P \Phi w_D^\star) \tag{2}$$

where $\Pi_D = \Phi^T (\Phi^T D \Phi)^{-1} \Phi^T D$ is a projection matrix weighted by the diagonal matrix $D \in \mathbb{R}^{n \times n}$. Rearranging terms gives the analytical solution

$$w_D^\star = \left(\Phi^T D (\Phi - \gamma P \Phi)\right)^{-1} \Phi^T D R. \tag{3}$$

Although we cannot expect to form this solution exactly when $P$ is unknown and too large to represent, we can approximate the solution via stochastic iteration (leading to the original TD algorithm), or via the least-squares TD (LSTD) algorithm, which forms the matrices

$$\hat{w}_D = \hat{A}^{-1} \hat{b}, \quad \hat{A} = \frac{1}{m} \sum_{i=1}^{m} \phi(s^{(i)}) \left(\phi(s^{(i)}) - \gamma \phi(s'^{(i)})\right), \quad \hat{b} = \frac{1}{m} \sum_{i=1}^{m} \phi(s^{(i)}) r^{(i)} \tag{4}$$

given a sequence of states, rewards, and next states $\{s^{(i)}, r^{(i)}, s'^{(i)}\}_{i=1}^{m}$ where $s^{(i)} \sim D$. When $D$ is not the stationary distribution of the Markov chain (i.e., we are employing off-policy sampling), then the original TD algorithm may diverge (LSTD will still be able to compute the TD fixed point in this case, but has a greater computational complexity of $O(k^2)$). Thus, there has been a great deal of work on developing $O(k)$ algorithms that are guaranteed to converge to the LSTD fixed point even in the case of off-policy sampling [16, 15].

We note that the above formulation avoids any explicit mention of a Markov Decision Process (MDP) or actual policies: rather, we just have tuples of the form $\{s, r, s'\}$ where $s$ is drawn from an arbitrary distribution but $s'$ still follows the "policy" we are trying to evaluate. This is a standard formulation for off-policy learning (see e.g. [16, Section 2]); briefly, the standard way to reach this setting from the typical notion of off-policy learning (acting according to one policy in an MDP, but evaluating another) is to act according to some original policy in an MDP, and then subsample only those actions that are immediately consistent with the policy of interest. We use the above notation as it avoids the need for any explicit notation of actions and still captures the off-policy setting completely.

### 2.1    Error bounds for the TD fixed point

Of course, in addition to the issue of convergence, there is the question as to whether we can say anything about the quality of the approximation at this fixed point. For the case of on-policy sampling, the answer here is an affirmative one, as formalized in the following theorem.

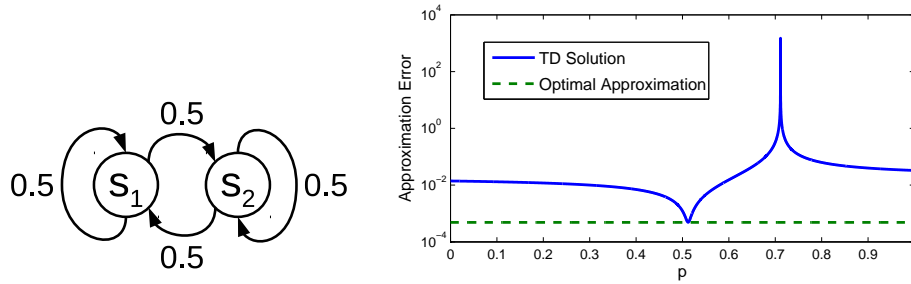

Figure 1: Counter example for off-policy TD learning: (left) the Markov chain considered for the counterexample; (right) the error of the TD estimate for different off-policy distributions (plotted on a log scale), along with the error of the optimal approximation.

**Theorem 1.** *(Tsitsiklis and Van Roy [17], Lemma 6) Let $w_D^\star$ be the unique solution to $\Phi w_D^\star = \Pi_D(R + \gamma P \Phi w_D^\star)$ where $D$ is the stationary distribution of $P$. Then*

$$\|\Phi w_D^\star - V\|_D \leq \frac{1}{1-\gamma} \|\Pi_D V - V\|_D. \tag{5}$$

Thus, for on-policy sampling with linear function approximation, not only does TD converge to its fixed point, but we can also bound the error of its approximation relative to $\|\Pi_D V - V\|_D$, the lowest possible approximation error for the class of function approximators.[1]

Since this theorem plays an integral role in the remainder of this paper, we want to briefly give the intuition of its proof. A fundamental property of Markov chains [17, Lemma 1] is that transition matrix $P$ is non-expansive in the $D$ norm when $D$ is the stationary distribution

$$\|Px\|_D \leq \|x\|_D, \ \forall x. \tag{6}$$

From this it can be shown that the Bellman operator is a $\gamma$-contraction in the $D$ norm and Theorem 1 follows. When $D$ is *not* the stationary distribution of the Markov chain, then (6) need not hold, and it remains to be seen what, if anything, can be said a priori about the TD fixed point in this situation.

## 3 An off-policy counterexample

Here we present a simple counter-example which shows, for general off-policy sampling, that the TD fixed point can be an arbitrarily poor approximator of the value function, even if the chosen bases can represent the true value function with low error. The same intuition has been presented previously [11]. though we here present a concrete numerical example for illustration.

**Example 1.** *Consider the two-state Markov chain shown in Figure 1, with transition probability matrix $P = (1/2)11^T$, discount factor $\gamma = 0.99$, and value function $V = [1 \ \ 1.05]^T$ (with $R = (I - \gamma P)V$ ). Then for any $\epsilon > 0$ and $C > 0$, there exists an off-policy distribution $D$ such that using bases $\Phi = [1 \ \ 1.05 + \epsilon]^T$ gives*

$$\|\Pi_D V - V\| \leq \epsilon, \ \ and \ \ \|\Phi w_D^\star - V\| \geq C. \tag{7}$$

*Proof.* (Sketch) The fact that $\|\Pi_D V - V\| \leq \epsilon$ is obvious from the choice of basis. To show that the TD error can be unboundedly large, let $D = \text{diag}(p, 1 - p)$; then, after some simplification, the TD solution is given analytically by

$$w_D^\star = \frac{-2961 + 4141p - 2820\epsilon + 2820p\epsilon}{-2961 + 4141p - 45240\epsilon + 84840p\epsilon - 40400\epsilon^2 + 40400p\epsilon^2} \tag{8}$$

which is infinite, $(1/w = 0)$, when

$$p = \frac{2961 + 45240\epsilon + 40400\epsilon^2}{4141 + 84840\epsilon + 40400\epsilon^2}. \tag{9}$$

Since this solution is in $(0, 1)$ for all epsilon, by choosing $p$ close to this value, we can make $w_D^\star$ arbitrarily large, which in turn makes the error of the TD estimate arbitrarily large. $\square$

Figure 1 shows a plot of $\|\Phi w^\star - V\|_2$ for the example above with $\epsilon = 0.001$, varying $p$ from 0 to 1. For $p \approx 0.715$ the error of the TD solution approaches infinity; the essential problem here is that when $D$ is not the stationary distribution of $P$, $A = \Phi^T D(\Phi - \gamma P\Phi)$ can become close to zero (or for the matrix case, one of its eigenvalues can become zero), and the TD value function estimate can grow unboundedly large. Thus, we argue that simple convergence for an off-policy algorithm is not a sufficient criterion for a good learning system, since even for a convergent algorithm the quality of the actual solution could be arbitrarily poor.

## 4   A convex characterization of valid off-policy distributions

Although it may seem as though the above example would imply that very little could be said about the quality of the TD fixed point under off-policy sampling, in this section we show that by imposing additional constraints on the sampling distribution, we can find a convex family of distributions for which it *is* possible to make guarantees.

To motivate the approach, we again note that error bounds for the on-policy TD algorithm follow from the Markov chain property that $\|Px\|_D \leq \|x\|_D$ for all $x$ when $D$ is the stationary distribution. However, finding a $D$ that satisfies this condition is no easier than computing the stationary distribution directly and thus is not a feasible approach. Instead, we consider a relaxed contraction property: that the transition matrix $P$ followed by a projection onto the bases will be non-expansive for any function already in the span of $\Phi$. Formally, we want to consider distributions $D$ for which

$$\|\Pi_D P\Phi w\|_D \leq \|\Phi w\|_D \tag{10}$$

for any $w \in \mathbb{R}^k$. This defines a *convex* set of distributions, since

$$\|\Pi_D P\Phi w\|_D^2 \leq \|\Phi w\|_D^2$$
$$\Leftrightarrow \quad w^T \Phi^T P^T D\Phi(\Phi^T D\Phi)^{-1}\Phi^T D\Phi(\Phi^T D\Phi)^{-1}\Phi D P\Phi^T w \leq w^T \Phi^T D\Phi w \tag{11}$$
$$\Leftrightarrow \quad w^T \left( \Phi^T P^T D\Phi(\Phi^T D\Phi)^{-1}\Phi D P\Phi^T - \Phi^T D\Phi \right) w \leq 0.$$

This holds for all $w$ if and only if[2]

$$\Phi^T P^T D\Phi(\Phi^T D\Phi)^{-1}\Phi D P\Phi^T - \Phi^T D\Phi \preceq 0 \tag{12}$$

which in turn holds if and only if[3]

$$F \equiv \left[ \begin{array}{cc} \Phi^T D\Phi & \Phi^T D P\Phi \\ \Phi^T P^T D\Phi & \Phi^T D\Phi \end{array} \right] \succeq 0 \tag{13}$$

This is a matrix inequality (LMI) in $D$, and thus describes a convex set. Although the distribution $D$ is too high-dimensional to optimize directly, analogous to LSTD, the $F$ matrix defined above *is* of a representable size ($2k \times 2k$), and can be approximated from samples. We will return to this point in the subsequent section, and for now will continue to use the notation of the true distribution $D$ for simplicity. The chief theoretical result of this section is that if we restrict our attention to off-policy distributions within this convex set, we can prove non-trivial bounds about the approximation error of the TD fixed point.

**Theorem 2.** *Let $w^\star$ be the unique solution to $\Phi w^\star = \Pi_D(R + \gamma P\Phi w^\star)$ where $D$ is any distribution satisfying (13). Further, let $D_\mu$ be the stationary distribution of $P$, and let $\bar{D} \equiv D^{-1/2}D_\mu^{1/2}$ Then[4]*

$$\|\Phi w_D^\star - V\|_D \leq \frac{1 + \gamma\kappa(\bar{D})}{1 - \gamma}\|\Pi_D V - V\|_D. \tag{14}$$

The bound here is of a similar form to the previously stated bound for on-policy TD, it bounds the error of the TD solution relative to the error of the best possible approximation, except for the additional $\gamma\kappa(\bar{D})$ term, which measures how much the chosen distribution deviates from the stationary distribution. When $D = D_\mu$, $\kappa(\bar{D}) = 1$, so we recover the original bound up to a constant factor. Even though the bound does include this term that depends on the distance from the stationary distribution, no such bound is possible for $D$ that do not satisfy the convex constraint (13), as illustrated by the previous counter-example.

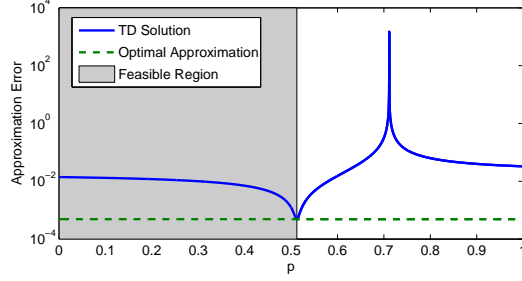

Figure 2: Counter example from Figure 1 shown with the set of all valid distributions for which $F \succeq 0$. Restricting the solution to this region avoids the possibility of the high error solution.

*Proof.* (of Theorem 2) By the triangle inequality and the definition of the TD fixed point,

$$
\begin{aligned}
\|\Phi w_D^\star - V\|_D &\le \|\Phi w_D^\star - \Pi_D V\|_D + \|\Pi_D V - V\|_D \\
&= \|\Pi_D(R + \gamma P \Phi w_D^\star) - \Pi_D(R + \gamma P V)\|_D + \|\Pi_D V - V\|_D \\
&= \gamma\|\Pi_D P \Phi w_D^\star - \Pi_D P V\|_D + \|\Pi_D V - V\|_D \\
&\le \gamma\|\Pi_D P \Phi w_D^\star - \Pi_D P \Pi_D V\|_D + \gamma\|\Pi_D P \Pi_D V - \Pi_D P V\|_D + \|\Pi_D V - V\|_D.
\end{aligned}
\tag{15}
$$

Since $\Pi_D V = \Phi \bar{w}$ for some $\bar{w}$, we can use the definition of our contraction $\|\Pi_D P \Phi w\|_D \le \|\Phi w\|_D$ to bound the first term as

$$
\|\Pi_D P \Phi w_D^\star - \Pi_D P \Pi_D V\|_D \le \|\Phi w_D^\star - \Pi_D V\|_D \le \|\Phi w_D^\star - V\|_D.
\tag{16}
$$

Similarly, the second term in (15) can be bounded as

$$
\|\Pi_D P \Pi_D V - \Pi_D P V\|_D \le \|P \Pi_D V - P V\|_D \le \|P\|_D \|\Pi_D V - V\|_D
\tag{17}
$$

where $\|P\|_D$ denotes the matrix norm $\|A\|_D \equiv \max_{\|x\|_D \le 1} \|Ax\|_D$. Substituting these bounds back into (15) gives

$$
(1 - \gamma)\|\Phi w_D^\star - V\|_D \le (1 + \gamma\|P\|_D)\|\Pi_D V - V\|_D,
\tag{18}
$$

so all the remains is to show that $\|P\|_D \le \kappa(\bar{D})$. To show this, first note that $\|P\|_{D_\mu} = 1$, since

$$
\max_{\|x\|_{D_\mu} \le 1} \|Px\|_{D_\mu} \le \max_{\|x\|_{D_\mu} \le 1} \|x\|_{D_\mu} = 1,
\tag{19}
$$

and for any nonsingular $D$,

$$
\|P\|_D = \max_{\|x\|_D \le 1} \|Px\|_D = \max_{\|y\|_2 \le 1} \sqrt{y^T D^{-1/2} P^T D P D^{-1/2} y} = \|D^{1/2} P D^{-1/2}\|_2.
\tag{20}
$$

Finally, since $D_\mu$ and $D$ are both diagonal (and thus commute),

$$
\begin{aligned}
\|D^{1/2} P D^{-1/2}\|_2 &= \|D_\mu^{-1/2} D^{1/2} D_\mu^{1/2} P D_\mu^{-1/2} D^{-1/2} D_\mu^{1/2}\|_2 \\
&\le \|D_\mu^{-1/2} D^{1/2}\|_2 \|D_\mu^{1/2} P D_\mu^{-1/2}\|_2 \|D^{-1/2} D_\mu^{1/2}\|_2 \\
&= \|D_\mu^{-1/2} D^{1/2}\|_2 \|D^{-1/2} D_\mu^{1/2}\|_2 = \kappa(\bar{D}) \qquad \square
\end{aligned}
$$

The final form of the bound can be quite loose of, course, as many of the steps involved in the proof used substantial approximations and discarded problem specific data (such as the actual $\|\Pi_D P\|_D$ term in favor of the generic $\kappa(\bar{D})$ term, for instance). This is in constrast to the previously mentioned work of Yu and Bertsekas [19] that uses these and similar terms to obtain much tigher, but data dependent, bounds. Indeed, applying a theorem from this work we can arrive as a slight improvement of the bound above [13], but the focus here is just on the general form and possibility of the bound.

Returning to the counter-example from the previous section, we can visualize the feasible region for which $F \succeq 0$, shown as the shaded portion in Figure 2, and so constraining the solution to this feasible region avoids the possibility of the high error solution. Moreover, in this example the optimal TD error occurs exactly at the point where $\lambda_{\min}(F) = 0$, so that projecting an off-policy distribution onto this set will give an optimal solution for initially infeasible distributions.

## 4.1 Estimation from samples

Returning to the issue of optimizing this distribution only using samples from the system, we note that analogous to LSTD, for samples $\{s^{(i)}, r^{(i)}, s'^{(i)}\}_{i=1}^{m}$

$$\hat{F} = \frac{1}{m} \sum_{i=1}^{m} \begin{bmatrix} \phi(s^{(i)})\phi(s^{(i)})^T & \phi(s^{(i)})\phi(s'^{(i)})^T \\ \phi(s'^{(i)})\phi(s^{(i)})^T & \phi(s^{(i)})\phi(s^{(i)})^T \end{bmatrix} \equiv \frac{1}{m} \sum_{i=1}^{m} \hat{F}_i \qquad (21)$$

will be an unbiased estimate of the LMI matrix $F$ (for a diagonal matrix $D$ given the our sampling distribution over $s^{(i)}$). Placing a weight $d_i$ on each sample, we could optimize the sum $\hat{F}(d) = \sum_{i=1}^{m} d_i \hat{F}_i$ and obtain a tractable optimization problem. However, optimizing these weights freely is not permissible, since this procedure allows us to choose $d_i \neq d_j$ even if $s^{(i)} = s^{(j)}$, which violates the weights in the original LMI. However, if we additionally require that $s^{(i)} = s^{(j)} \Rightarrow d_i = d_j$ (or more appropriately for continuous features and states, for example that $\|d_i - d_j\| \to 0$ as $\|\phi(s^{(i)}) - \phi(s^{(j)})\| \to 0$ according to some norm) then we are free to optimize over these empirical distribution weights. In practice, we want to constrain this distribution in a manner commensurate with the complexity of the feature space and the number of samples. However, determining the best such distributions to use in practice remains an open problem for future work in this area.

Finally, since many empirical distributions satisfy $\hat{F}(d) \succeq 0$, we propose to "project" the empirical distribution onto this set by minimizing the KL divergence between the observed and optimized distributions, subject to the constraint that $\hat{F}(d) \succeq 0$. Since this constraint is guaranteed to hold at the stationary distribution, the intuition here is that by moving closer to this set, we will likely obtain a better solution. Formally, the final optimization problem, which we refer to as the TD-DO method (Temporal Difference Distribution Optimization), is given by

$$\min_{d} \sum_{i=1}^{m} -\hat{p}_i \log d_i \quad \text{s.t.} \quad , \quad 1^T d = 0, \quad \hat{F}(d) \succeq 0, \quad d \in \mathcal{C}. \qquad (22)$$

where $\mathcal{C}$ is some convex set that respects the metric constraints described above. This is a convex optimization problem in $d$, and thus can be solved efficiently, though off-the-shelf solvers can perform quite poorly, especially for large dimension $m$.

## 4.2 Efficient Optimization

Here we present a first-order optimization method based on solving the dual of (22). By properly exploiting the decomposability of the objective and low-rank structure of the dual problem, we develop an iterative optimization method where each gradient step can be computed very efficiently. The presentation here is necessarily brief due to space constraints, but we also include a longer description and an implementation of the method in the supplementary material. For simplicity we present the algorithm ignoring the constraint set $\mathcal{C}$, though we discuss possible additonal constraints briefly in supplementary material.

We begin by forming the Lagrangian of (22), introducing Lagrange multipliers $Z \in \mathbb{R}^{2k \times 2k}$ for the constraint $\hat{F}(d) \succeq 0$ and $\nu \in \mathbb{R}$ for the constraint $1^T d = 1$. This leads to the dual optimization problem

$$\max_{Z \succeq 0, \nu} \min_{d} \left\{ -\sum_{i=1}^{m} \hat{p}_i \log d_i - \text{tr}(Z^T \hat{F}(d)) + \nu(1^T d - 1) \right\}. \qquad (23)$$

Treating $Z$ as fixed, we maximize over $\nu$ and minimize over $d$ in (23) using an equality-constrained, feasible start Newton method [2, pg 528]. Since the objective is separable over the $d_i$'s the Hessian matrix is diagonal, and the Newton step can be computed in $O(m)$ time; furthermore, since we solve this subproblem for each update of dual variables $Z$, we can warm-start Newton's method from previous solutions, leading to a number of Newton steps that is virtually constant in practice.

Considering now the maximization over $Z$, the gradient of

$$g(Z) \equiv \left\{ \sum_i -\hat{p}_i \log d_i^\star(Z) - \text{tr}Z^T \hat{F}(d^\star(Z)) + \nu^\star(Z)(1^T d^\star(Z) - 1) \right\} \qquad (24)$$

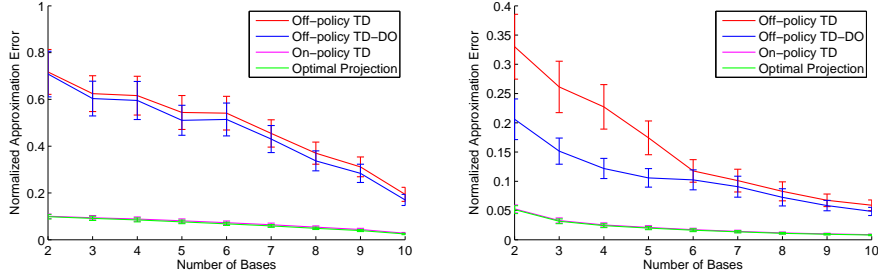

Figure 3: Average approximation error of the TD methods, using different numbers of bases functions, for the random Markov chain (left) and diffusion chain (right).

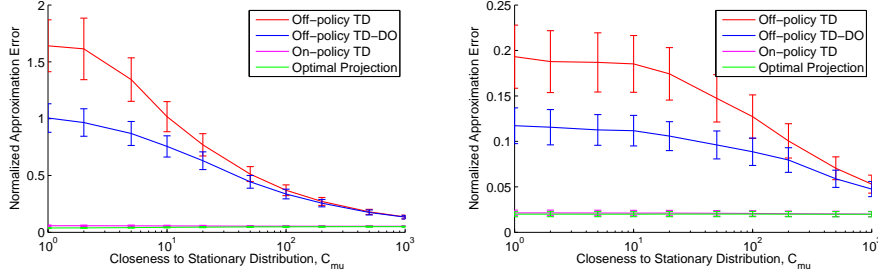

Figure 4: Average approximation error, using off-policy distributions closer or further from the stationary distribution (see text) for the random Markov chain (left) and diffusion chain (right).

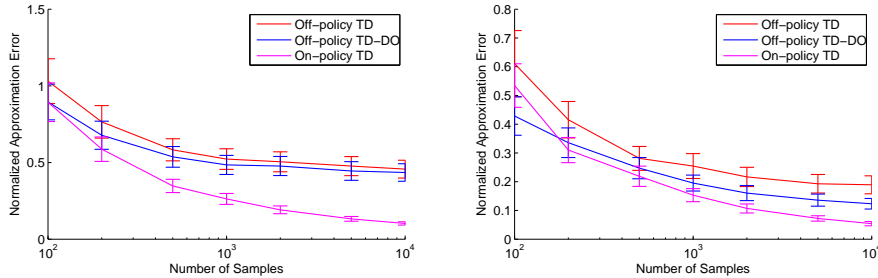

Figure 5: Average approximation error for TD methods computed via sampling, for different numbers of samples, for random Markov chain (left) and diffusion chain (right).

is given simply by $\nabla_Z g(Z) = -\hat{F}(d^\star(Z))$. We then exploit the fact that we expect $Z$ to typically be *low-rank*: by the KKT conditions for a semidefinite program $\hat{F}(d)$ and $Z$ will have complementary ranks, and since we expect $\hat{F}(d)$ to be nearly full rank at the solution, we factor $Z = YY^T$ for $Y \in \mathbb{R}^{k \times p}$ with $p \ll k$. Although this is now a non-convex problem, local optimization of this objective is still guaranteed to give a global solution to the original semidefinite problem, provided we choose the rank of $Y$ to be sufficient to represent the optimal solution [5]. The gradient of this transformed problem is $\nabla_Z g(YY^T) = -2\hat{F}(d)Y$, which can be computed in time $O(mkp)$ since each $\hat{F}_i$ term is a low-rank matrix, and we optimize the dual objective via an off-the-shelf LBFGS solver [12, 14]. Though it is difficult to bound $p$ aprirori, we can check after the solution that our chosen value was sufficient for the global solution, and we have found that very low values ($p = 1$ or $p = 2$) were sufficient in our experiments.

## 5    Experiments

Here we present simple simulation experiments illustrating our proposed approach; while the evaluation is of course small scale, the results highlight the potential of TD-DO to improve TD algorithms both practically as well as theoretically. Since the benefits of the method are clearest in terms of

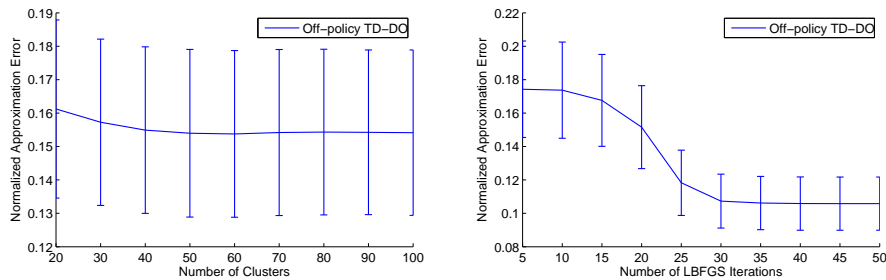

Figure 6: (Left) Effect of the number of clusters for sample-based learning on diffusion chain, (Right) performance of algorithm on diffusion chain versus number of LBFGS iterations

the mean performance over many different environments we focus on randomly generated Markov chains of two types: a random chain and a diffusion process chain.[5]

Figure 3 shows the average approximation error of the different algorithms with differing numbers of basis function, over 1000 domains. In this and all experiments other than those evaluating the effect of sampling, we use the full $\Phi$ and $P$ matrices to compute the convex set, so that we are evaluating the performance of the approach in the limit of large numbers of samples. We evaluate the approximation error $\|\hat{V} - V\|_D$ where $D$ is the off-policy sampling distribution (so as to be as favorable as possible to off-policy TD). In all cases the TD-DO algorithm improves upon the off-policy TD, though the degree of improvement can vary from minor to quite significant.

Figure 4 shows a similar result for varying the closeness of the sampling distribution to the stationary distribution; in our experiments, the off-policy distribution is sampled according to $D \sim \mathrm{Dir}(1 + C_\mu \mu)$ where $\mu$ denotes the stationary distribution. As expected, the off-policy approaches perform similarly for larger $C_\mu$ (approaching the stationary distribution), with TD-DO having a clear advantage when the off-policy distribution is far from the stationary distribution.

In Figure 5 we consider the effect of sampling on the algorithms. For these experiments we employ a simple clustering method to compute a distribution over states $d$ that respects the fact that $\phi(s^{(i)}) = \phi(s^{(j)}) \Rightarrow d_i = d_j$: we group the sampled states into $k$ clusters via $k$-means clustering on the feature vectors, and optimize over the reduced distribution $d \in \mathbb{R}^k$. In Figure 6 we vary the number of clusters $k$ for the sampled diffusion chain, showing that the algorithm is robust to a large number of different distributional representations; we also show the performance of our method varying the number of LBFGS iterations, illustrating that performance generally improves monotonically.

## 6 Conclusion

The fundamental idea we have presented in this paper is that by considering a convex subset of off-policy distributions (and one which can be computed efficiently from samples), we can provide performance guarantees for the TD fixed point. While we have focused on presenting error bounds for the analytical (infinite sample) TD fixed point, a huge swath of problems in TD learning arise from this same off-policy issue: the convergence of the original TD method, the ability to find the $\ell_1$ regularized TD fixed point [6], the on-policy requirement of the finite sample analysis of LSTD [8], and the convergence of TD-based policy iteration algorithms [7]. Although left for future work, we suspect that the same techniques we present here can also be extending to these other cases, potentially providing a wide range of analogous results that still apply under off-policy sampling.

**Acknowledgements.** We thank the reviewers for helpful comments and Bruno Scherrer for pointing out a potential improvement to the error bound. J. Zico Kolter is supported by an NSF CI Fellowship.

## Footnotes

[1]The approximation factor can be sharpened to $\frac{1}{\sqrt{1-\gamma^2}}$ in some settings [18], though the analysis does not carry over to our off-policy case, so we present here the simpler version.

[2] $A \preceq 0$ ($A \succeq 0$) denotes that $A$ is negative (positive) semidefinite.

[3] Using the Schur complement property that $\left[ \begin{array}{cc} A & B \\ B^T & C \end{array} \right] \succeq 0 \iff B^T AB - C \preceq 0$ [2, pg 650-651].

[4] $\kappa(A)$ denotes the condition number of $A$, the ratio of the singular values $\kappa(A) = \sigma_{\max}(A)/\sigma_{\min}(A)$.

[5]Experimental details: For the random Markov Chain rows of $P$ are drawn IID from a Dirichlet distribution, and the reward and bases are random normal, with $|S| = 11$. For the diffusion-based chain, we sample $|S| = 100$ points from a 2D unit cube $x_i \in [0, 1]^2$ and set $p(s' = j | s = i) \propto \exp(-\|x_i - x_j\|^2 / (2\sigma^2))$ for bandwidth $\sigma = 0.4$. Similarly, rewards are sampled from a zero-mean Gaussian Process with covariance $K_{ij} = \exp(-\|x_i - x_j\|^2 / (2\sigma^2))$, and for basis vectors we use the principle eigenvectors of $\mathrm{Cov}(V) = \mathbf{E}[(I - \gamma P) R R^T (I - \gamma P)^T] = (I - \gamma P) K (I - \gamma P)^T$, which are the optimal bases for representing value functions (in expectation). Some details of the domains are omitted due to space constraints, but MATLAB code for all the experiments is included in the supplementary files.

## References

[1] L. C. Baird. Residual algorithms: Reinforcement learning with function approximation. In *Proceedings of the International Conference on Machine Learning*, 1995.

[2] S. Boyd and L. Vandenberghe. *Convex Optimization*. Cambridge University Press, 2004.

[3] P. Dayan. The convergence of TD($\lambda$) for general $\lambda$. *Machine Learning*, 8(3–4), 1992.

[4] T. Jaakkola, M. I. Jordan, and S. P. Singh. On the convergence of stochastic iterative dynamic programming algorithms. *Neural Computation*, 6(6), 1994.

[5] M. Journee, F. Bach, P.A. Absil, and R. Sepulchre. Low-rank optimization on the cone of positive semidefinite matrices. *SIAM Journal on Optimization*, 20(5):2327–2351, 2010.

[6] J.Z. Kolter and A.Y. Ng. Regularization and feature selection in least-squares temporal difference learning. In *Proceedings of the International Conference on Machine Learning*, 2009.

[7] M. G. Lagoudakis and R. Parr. Least-squares policy iteration. *Journal of Machine Learning Research*, 4:1107–1149, 2003.

[8] A. Lazaric, M. Ghavamzadeh, and R. Munos. Finite-sample analysis of LSTD. In *Proceedings of the International Conference on Machine Learning*, 2010.

[9] H.R. Maei and R.S. Sutton. GQ($\lambda$): A general gradient algorithm for temporal-difference prediction learning with eligibility traces. In *Proceedings of the Third Conference on Artificial General Intelligence*, 2010.

[10] H.R. Maei, Cs. Szepesvari, S. Bhatnagar, and R.S. Sutton. Toward off-policy learning control with function approximation. In *Proceedings of the International Conference on Machine Learning*, 2010.

[11] R. Munos. Error bounds for approximate policy iteration. In *Proceedings of the International Conference on Machine Learning*, 2003.

[12] J. Nocedal and S.J. Wright. *Numerical Optimization*. Springer, 1999.

[13] B. Scherrer. Personal communication, 2011.

[14] M. Schmidt. minfunc, 2005. Available at `http://www.cs.ubc.ca/~schmidtm/Software/minFunc.html`.

[15] R.S. Sutton, H.R. Maei, D. Precup, S. Bhatnagar, D. Silver, Cs. Szepesvari, and E. Wiewiora. Fast gradient-descent methods for temporal-difference learning with linear function approximation. In *Proceedings of the International Conference on Machine Learning*, 2009.

[16] R.S. Sutton, Cs. Szepesvari, and H.R. Maei. A convergent O(n) algorithm for off-policy temporal-different learning with linear function approximation. In *Advances in Neural Information Processing*, 2008.

[17] J.N. Tsitsiklis and B. Van Roy. An analysis of temporal-difference learning with function approximation. *IEEE Transactions and Auotomatic Control*, 42:674–690, 1997.

[18] J.N. Tsitsiklis and B. Van Roy. Average cost temporal difference learning. *Automatica*, 35(11):1799–1808, 1999.

[19] H. Yu and D. P. Bertsekas. Error bounds for approximations from projected linear equations. *Mathematics of Operations Research*, 35:306–329, 2010.

